# Kernel Methods for Implicit Surface Modeling

**Bernhard Schölkopf**[†]**, Joachim Giesen**[+][*]**& Simon Spalinger**[+]
[†] Max Planck Institute for Biological Cybernetics, 72076 Tübingen, Germany
`bernhard.schoelkopf@tuebingen.mpg.de`
[+] Department of Computer Science, ETH Zürich, Switzerland
`giesen@inf.ethz.ch,spsimon@inf.ethz.ch`

## Abstract

We describe methods for computing an implicit model of a hypersurface that is given only by a finite sampling. The methods work by mapping the sample points into a reproducing kernel Hilbert space and then determining regions in terms of hyperplanes.

## 1  Introduction

Suppose we are given a finite sampling (in machine learning terms, training data) $x_1, \ldots, x_m \in \mathcal{X}$, where the domain $\mathcal{X}$ is some hypersurface in Euclidean space $\mathbb{R}^d$. The case $d = 3$ is especially interesting since these days there are many devices, e.g., laser range scanners, that allow the acquisition of point data from the boundary surfaces of solids. For further processing it is often necessary to transform this data into a continuous model. Today the most popular approach is to add connectivity information to the data by transforming them into a triangle mesh (see [4] for an example of such a transformation algorithm). But recently also implicit models, where the surface is modeled as the zero set of some sufficiently smooth function, gained some popularity [1]. They bear resemblance to level set methods used in computer vision [6]. One advantage of implicit models is that they easily allow the derivation of higher order differential quantities such as curvatures. Another advantage is that an inside-outside test, i.e., testing whether a query point lies on the bounded or unbounded side of the surface, boils down to determining the sign of a function-evaluation at the query point. Inside-outside tests are important when one wants to intersect two solids.

The goal of this paper is, loosely speaking, to find a function which takes the value zero on a surface which

(1) contains the training data and

(2) is a "reasonable" implicit model of $\mathcal{X}$.

To capture properties of its shape even in the above general case, we need to exploit some structure on $\mathcal{X}$. In line with a sizeable amount of recent work on kernel methods [11], we assume that this structure is given by a *(positive definite) kernel*, i.e., a real valued function

---

[*]Partially supported by the Swiss National Science Foundation under the project "Non-linear manifold learning".

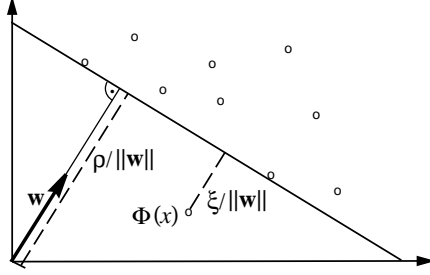

Figure 1: In the 2-D toy example depicted, the hyperplane $\langle \mathbf{w}, \Phi(x) \rangle = \rho$ separates all but one of the points from the origin. The outlier $\Phi(x)$ is associated with a slack variable $\xi$, which is penalized in the objective function (4). The distance from the outlier to the hyperplane is $\xi / \|\mathbf{w}\|$; the distance between hyperplane and origin is $\rho / \|\mathbf{w}\|$. The latter implies that a small $\|\mathbf{w}\|$ corresponds to a large margin of separation from the origin.

$k$ on $\mathcal{X} \times \mathcal{X}$ which can be expressed as

$$k(x, x') = \langle \Phi(x), \Phi(x') \rangle \tag{1}$$

for some map $\Phi$ into a Hilbert space $\mathcal{H}$. The space $\mathcal{H}$ is the *reproducing kernel Hilbert space (RKHS)* associated with $k$, and $\Phi$ is called its *feature map*. A popular example, in the case where $\mathcal{X}$ is a normed space, is the Gaussian (where $\sigma > 0$)

$$k(x, x') = \exp\left( -\frac{\|x - x'\|^2}{2\,\sigma^2} \right). \tag{2}$$

The advantage of using a positive definite kernel as a similarity measure is that it allows us to construct geometric algorithms in Hilbert spaces.

## 2    Single-Class SVMs

Single-class SVMs were introduced [8, 10] to estimate quantiles $C \approx \{x \in \mathcal{X} | f(x) \in [\rho, \infty[\}$ of an unknown distribution $P$ on $\mathcal{X}$ using kernel expansions. Here,

$$f(x) = \sum_i \alpha_i k(x_i, x) - \rho, \tag{3}$$

where $x_1, \dots, x_m \in \mathcal{X}$ are unlabeled data generated i.i.d. according to $P$. The single-class SVM approximately computes the smallest set $C \in \mathcal{C}$ containing a specified fraction of all training examples, where smallness is measured in terms of the norm in the RKHS $\mathcal{H}$ associated with $k$, and $\mathcal{C}$ is the family of sets corresponding to half-spaces in $\mathcal{H}$. Depending on the kernel, this notion of smallness will coincide with the intuitive idea that the quantile estimate should not only contain a specified fraction of the training points, but it should also be sufficiently smooth so that the same is approximately true for previously unseen points sampled from $P$.

Let us briefly describe the main ideas of the approach. The training points are mapped into $\mathcal{H}$ using the feature map $\Phi$ associated with $k$, and then it is attempted to separate them from the origin with a large margin by solving the following quadratic program: for $\nu \in (0, 1]$,[1]

$$\underset{\mathbf{w} \in \mathcal{H}, \boldsymbol{\xi} \in \mathbb{R}^m, \rho \in \mathbb{R}}{\text{minimize}} \quad \frac{1}{2} \|\mathbf{w}\|^2 + \frac{1}{\nu m} \sum_i \xi_i - \rho \tag{4}$$

$$\text{subject to} \quad \langle \mathbf{w}, \Phi(x_i) \rangle \geq \rho - \xi_i, \ \ \xi_i \geq 0. \tag{5}$$

Since non-zero slack variables $\xi_i$ are penalized in the objective function, we can expect that if $\mathbf{w}$ and $\rho$ solve this problem, then the decision function, $f(x) = \text{sgn}\left( \langle \mathbf{w}, \Phi(x) \rangle - \rho \right)$ will

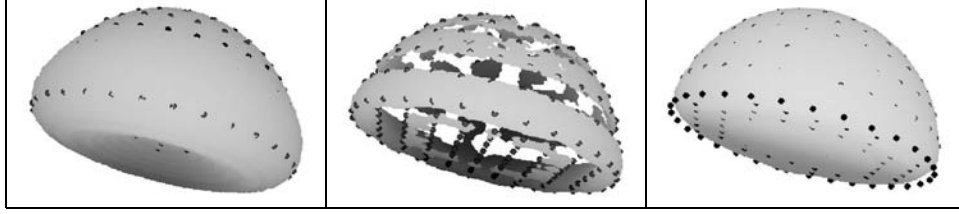

Figure 2: Models computed with a single class SVM using a Gaussian kernel (2). The three examples differ in the value chosen for $\sigma$ in the kernel - a large value (0.224 times the diameter of the hemisphere) in the left figure and a small value (0.062 times the diameter of the hemisphere) in the middle and right figure. In the right figure also non-zero slack variables (outliers) were allowed. Note that that the outliers in the right figure correspond to a sharp feature (non-smoothness) in the original surface.

equal 1 for most examples $x_i$ contained in the training set,[2] while the regularization term $\|\mathbf{w}\|$ will still be small. For an illustration, see Figure 1. The trade-off between these two goals is controlled by a parameter $\nu$.

One can show that the solution takes the form

$$f(x) = \text{sgn}\left(\sum_i \alpha_i k(x_i, x) - \rho\right), \tag{6}$$

where the $\alpha_i$ are computed by solving the dual problem,

$$\underset{\boldsymbol{\alpha} \in \mathbb{R}^m}{\text{minimize}} \quad \frac{1}{2} \sum_{ij} \alpha_i \alpha_j k(x_i, x_j) \tag{7}$$

$$\text{subject to} \quad 0 \leq \alpha_i \leq \frac{1}{\nu m} \text{ and } \sum_i \alpha_i = 1. \tag{8}$$

Note that according to (8), the training examples contribute with nonnegative weights $\alpha_i \geq 0$ to the solution (6). One can show that asymptotically, a fraction $\nu$ of all training examples will have strictly positive weights, and the rest will be zero (the "$\nu$-property").

In our application we are not primarily interested in a decision function itself but in the boundaries of the regions in input space defined by the decision function. That is, we are interested in $f^{-1}(0)$, where $f$ is the kernel expansion (3) and the points $x_1, \ldots, x_m \in \mathcal{X}$ are sampled from some unknown hypersurface $\mathcal{X} \subset \mathbb{R}^d$. We want to consider $f^{-1}(0)$ as a model for $\mathcal{X}$. In the following we focus on the case $d = 3$. If we assume that the $x_i$ are sampled without noise from $\mathcal{X}$ – which for example is a reasonable assumption for data obtained with a state of the art 3d laser scanning device – we should set the slack variables in (4) and (5) to zero. In the dual problem this results in removing the upper constraints on the $\alpha_i$ in (8). Note that sample points with non-zero slack variable cannot be contained in $f^{-1}(0)$. But also sample points whose image in feature space lies above the optimal hyperplane are not contained in $f^{-1}(0)$ (see Figure 1) — we will address this in the next section. It turns out that it is useful in practice to allow non-zero slack variables, because they prevent $f^{-1}(0)$ from decomposing into many connected components (see Figure 2 for an illustration).

In our experience, one can ensure that the images of all sample points in feature space lie close to (or on) the optimal hyperplane can be achieved by choosing $\sigma$ in the Gaussian

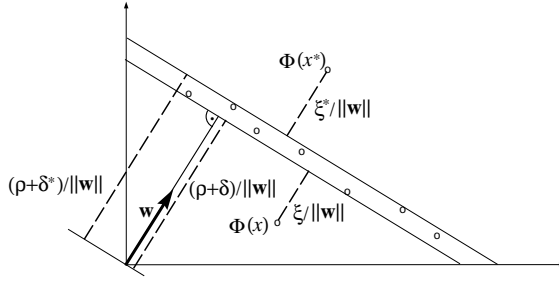

Figure 3: Two parallel hyperplanes $\langle \mathbf{w}, \Phi(x) \rangle = \rho + \delta^{(*)}$ enclosing all but two of the points. The outlier $\Phi(x^{(*)})$ is associated with a slack variable $\xi^{(*)}$, which is penalized in the objective function (9).

kernel (2) such that the Gaussians in the kernel expansion (3) are highly localized. However, highly localized Gaussians are not well suited for interpolation — the implicit surface decomposes into several components. Allowing outliers mitigates the situation to a certain extent. Another way to deal with the problem is to further restrict the optimal region in feature space. In the following we will pursue the latter approach.

## 3   Slab SVMs

A richer class of solutions, where some of the weights can be negative, is obtained if we change the geometric setup. In this case, we estimate a region which is a slab in the RKHS, i.e., the area enclosed between two parallel hyperplanes (see Figure 3).

To this end, we consider the following modified program:[3]

$$\underset{\mathbf{w} \in \mathcal{H}, \boldsymbol{\xi}^{(*)} \in \mathbb{R}^m, \rho \in \mathbb{R}}{\text{minimize}} \quad \frac{1}{2} \|\mathbf{w}\|^2 + \frac{1}{\nu m} \sum_i (\xi_i + \xi_i^*) - \rho \tag{9}$$

$$\text{subject to} \quad \delta - \xi_i \leq \langle \mathbf{w}, \Phi(x_i) \rangle - \rho \leq \delta^* + \xi_i^* \tag{10}$$

$$\text{and} \quad \xi_i^{(*)} \geq 0. \tag{11}$$

Here, $\delta^{(*)}$ are fixed parameters. Strictly speaking, one of them is redundant: one can show that if we subtract some offset from both, then we obtain the same overall solution, with $\rho$ changed by the same offset. Hence, we can generally set one of them to zero, say, $\delta = 0$.

Below we summarize some relationships of this convex quadratic optimization problem to known SV methods:

1. For $\delta = 0$ and $\delta^* = \infty$ (i.e., no upper constraint), we recover the single-class SVM (4)–(5).

2. If we drop $\rho$ from the objective function and set $\delta = -\varepsilon$, $\delta^* = \varepsilon$ (for some fixed $\varepsilon \geq 0$), we obtain the $\varepsilon$-insensitive support vector regression algorithm [11], for a data set where all output values $y_1, \ldots, y_m$ are zero. Note that in this case, the solution is trivial, $\mathbf{w} = 0$. This shows that the $\rho$ in our objective function plays an important role.

3. For $\delta = \delta^* = 0$, the term $\sum_i (\xi_i + \xi_i^*)$ measures the distance of the point $\Phi(x_i)$ from the hyperplane $\langle \mathbf{w}, \Phi(x_i) \rangle - \rho = 0$ (up to a scaling of $\|\mathbf{w}\|$). If $\nu$ tends to zero, this term will dominate the objective function. Hence, in this case, the solution will be a hyperplane that approximates the data well in the sense that the points lie close to it in the RKHS norm.

From the following constraints and Lagrange multipliers

$$\xi_i - \delta + \langle \mathbf{w}, \Phi(x_i) \rangle - \rho \geq 0, \qquad \alpha_i \geq 0 \tag{12}$$

$$\xi_i^* + \delta^* + \rho - \langle \mathbf{w}, \Phi(x_i) \rangle \geq 0, \qquad \alpha_i^* \geq 0 \tag{13}$$

$$\xi_i^{(*)} \geq 0, \qquad \beta_i^{(*)} \geq 0 \tag{14}$$

we derive the Lagrangian dual optimization problem of (9) - (11):[4]

$$\underset{\boldsymbol{\alpha} \in \mathbb{R}^m}{\text{minimize}} \quad \frac{1}{2} \sum_{ij} (\alpha_i - \alpha_i^*)(\alpha_j - \alpha_j^*) k(x_i, x_j) - \delta \sum_i \alpha_i + \delta^* \sum_i \alpha_i^* \tag{15}$$

$$\text{subject to} \quad 0 \leq \alpha_i^{(*)} \leq \frac{1}{\nu m} \tag{16}$$

$$\text{and} \quad \sum_i (\alpha_i - \alpha_i^*) = 1, \tag{17}$$

Note that for $\delta = \delta^*$, we can simplify the optimization problem using the transformation $\alpha^{new} = \alpha - \alpha^*$. For $\delta = \delta^* = 0$, we thus obtain the single-class SVM (7) with the modified box constraint $-\frac{1}{\nu m} \leq \alpha_i^{new} \leq \frac{1}{\nu m}$.

The dual problem can be solved using standard quadratic programming packages. The offset $\rho$ can be computed from the value of the corresponding variable in the double dual, or using the Karush-Kuhn-Tucker (KKT) conditions, just as in other support vector methods. Once this is done, we can evaluate for each test point $x$ whether it satisfies $\delta \leq \langle \mathbf{w}, \Phi(x) \rangle - \rho \leq \delta^*$. In other words, we have an implicit description of the region in input space that corresponds to the region in between the two hyperplanes in the RKHS. For $\delta = \delta^*$, this is a single hyperplane, corresponding to a hypersurface in input space.[5] To compute this surface we use the kernel expansion

$$\langle \mathbf{w}, \Phi(x) \rangle = \sum_i (\alpha_i - \alpha_i^*) k(x_i, x). \tag{18}$$

**Support Vectors and Outliers** In our discussion of single class SVMs for surface modeling we already mentioned that we aim for many support vectors (as we want most training points to lie on the surface) and that outliers might represent features like certain singularities in the original hypersurface.

Here we analyze how the parameter $\nu$ influences the SVs and outliers. To this end, we introduce the following shorthands for the sets of SV and outlier indices:

$$SV \quad := \quad \{i \mid \langle \mathbf{w}, \Phi(x_i) \rangle - \rho - \delta \leq 0\} \tag{19}$$

$$SV^* \quad := \quad \{i \mid \langle \mathbf{w}, \Phi(x_i) \rangle - \rho - \delta^* \geq 0\} \tag{20}$$

$$OL^{(*)} \quad := \quad \{i \mid \xi_i^{(*)} > 0\} \tag{21}$$

It is clear from the primal optimization problem that for all $i$, $\xi_i > 0$ implies $\langle \mathbf{w}, \Phi(x_i) \rangle - \rho - \delta < 0$ (and likewise, $\xi_i^* > 0$ implies $\langle \mathbf{w}, \Phi(x_i) \rangle - \rho - \delta^* > 0$), hence $OL^{(*)} \subset SV^{(*)}$. The difference of the SV and OL sets are those points that lie precisely on the boundaries of the constraints.[6] Below, $|A|$ denotes the cardinality of the set $A$.

**Proposition 1** *The solution of (9)–(11) satisfies*

$$\frac{|SV|}{m} - \frac{|OL^*|}{m} \geq \nu \geq \frac{|OL|}{m} - \frac{|SV^*|}{m}. \qquad (22)$$

The proof is analogous to the one of the "$\nu$-property" for standard SVMs, cf. [8]. Due to lack of space, we skip it, and instead merely add the following observations:

1. The above statements are not symmetric with respect to exchanging the quantities with asterisks and their counterparts without asterisk. This is due to the sign of $\rho$ in the primal objective function. If we used $+\rho$ rather than $-\rho$, we would obtain almost the same dual, the only difference being that the constraint (17) would have a "$-1$" on the right hand side. In this case, the role of the quantities with and without asterisks would be reversed in Proposition 1.

2. The $\nu$-property of single class SVMs is obtained as the special case where $OL^* = SV^* = \emptyset$.

3. Essentially, if we require that the distribution has a density w.r.t. the Lebesgue measure, and that $k$ is analytic and non-constant (cf. [8, 9]), it can be shown that asymptotically, the two inequalities in the proposition become equalities with probability 1.

**Implementation**   On larger problems, solving the dual with standard QP solvers becomes too expensive (scaling with $m^3$). For this case, we can use decomposition methods. The adaptation of known decomposition methods to the present case is straightforward, noticing that the dual of the standard $\varepsilon$-SV regression algorithm [11] becomes almost identical to the present dual if we set $\varepsilon = (\delta^* - \delta)/2$ and $y_i = -(\delta^* + \delta)/2$ for all $i$. The only difference is that in our case, there is a "1" in (17), whereas in the SVR case, we would have a "0". As a consequence, we have to change the initialization of the optimization algorithm to ensure that we start with a feasible solution. As an optimizer, we used a modified version of libSVM [2].

**Experimental Results**   In all our experiments we used a Gaussian kernel (2). To render the implicit surfaces, i.e., the zero-set $f^{-1}(0)$, we generated a triangle mesh that approximates it. To compute the mesh we used an adaptation of the *marching cubes* algorithm [5] which is a standard technique to transform an implicitly given surfaces into a mesh. The most costly operations in the marching cubes algorithm are evaluations of the kernel expansion (18). To reduce the number of these evaluations we implemented a surface following technique that exploits the fact that we know quite some sample points on the surface, namely the support vectors.[7] Some results can be seen in Figure 4.

Our experiments indicate a nice geometric interpretation of negative coefficients $\alpha_i - \alpha_i^*$. It seems that negative coefficients correspond to concavities in the original model. The coefficients seem well suited to extract shape features from the sample point set, e.g., the detection of singularities like sharp edges or feature lines — which is an important topic in computer graphics [7].

We also tried a multi-scale approach. In this approach at first a rough model is computed from ten percent of the sample points using a slab SVM. For the remaining 90% of the sample points we compute the residual values, i.e., we evaluate the kernel expansion (18) at the sample points. Finally we use support vector regression (SVR) and the residual values to derive a new kernel expansion (using a smaller kernel width) whose zero set we use as our surface model. An example how this approach works can be seen in Figure 5.

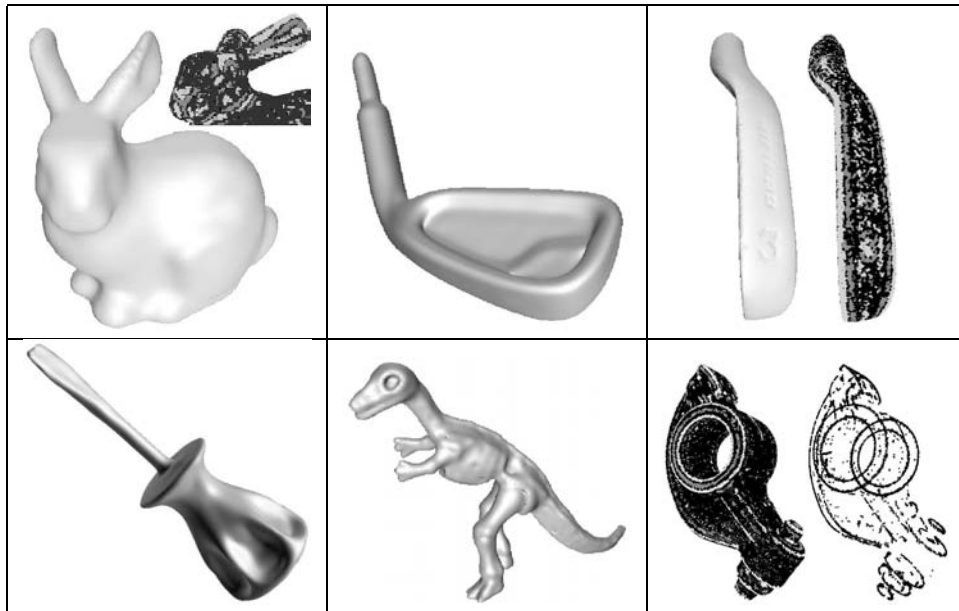

Figure 4: First row: Computing a model of the Stanford bunny (35947 points) and of a golf club (16864 points) with the slab SVM. The close up of the ears and nose of the bunny shows the sample points colored according to the coefficients $\alpha_i - \alpha_i^*$. Dark gray points have negative coefficients and light gray points positive ones. In the right figure we show the bottom of the golf club model. The model on the left of this figure was computed with a different method [4]. Note that with this method fine details like the figure three become visible. Such details get leveled out by the limited resolution of the marching cubes method. However the information about these details is preserved and detected in the SVM solution, as can be seen from the color coding. Second row: In the left and in the middle figure we show the results of the slab SVM method on the screwdriver model (27152 points) and the dinosaur model (13990 points), respectively. In the right figure a color coding of the coefficients for the rockerarm data set (40177 points) is shown. Note that we can extract sharp features from this data set by filtering the coefficients according to some threshold.

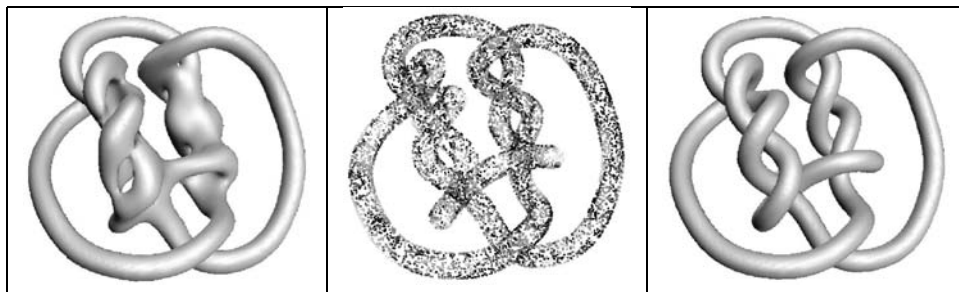

Figure 5: First row: The multi-scale approach applied to a knot data set (10000 points). The blobby support surface (left figure) was computed from 1000 randomly chosen sample points with the slab SVM. In the middle we show a color coding of the residual values of all sample points (cf. http://books.nips.cc for color images). In the right figure we show the surface that we get after applying support vector regression using the residual values.

## 4 Discussion and Outlook

An approximate description of the data as the zero set of a function can be useful as a compact representation of the data. It could potentially also be employed in other tasks where models of the data are useful, such as denoising and image super-resolution. We therefore consider it worthwhile to explore the algorithmic aspects of implicit surface estimation in more depth, including the study of regression based approaches.

Some acquisition devices do not only provide us with points from a surface embedded in $\mathbb{R}^3$, but also with the normals at these points. Using methods similar to the ones in [3], it should be possible to integrate such additional information into our approach. We expect that it will improve the quality of the computed models in the sense that even more geometric details are preserved.

A feature of our approach is that its complexity depends only marginally on the dimension of the input space (in our examples this was three). Thus the approach should work also well for hypersurfaces in higher dimensional input spaces. From an applications point of view hypersurfaces might not be as interesting as manifolds of higher co-dimension. It would be interesting to see if our approach can be generalized to handle also this situation.

**Acknowledgment**   We thank Chih-Jen Lin for help with libSVM. The bunny data were taken from the Stanford 3d model repository. The screwdriver, dinosaur and rockerarm data were taken from the homepage of Cyberware Inc. Thanks to Koby Crammer, Florian Steinke, and Christian Walder for useful discussion.

## Footnotes

[1] Here and below, bold face greek character denote vectors, e.g., $\boldsymbol{\xi} = (\xi_1, \dots, \xi_m)^\top$, and indices $i, j$ by default run over $1, \dots, m$.

[2]We use the convention that sgn $(z)$ equals 1 for $z \geq 0$ and $-1$ otherwise.

[3]Here and below, the superscript $(*)$ simultaneously denotes the variables with and without asterisk, e.g., $\boldsymbol{\xi}^{(*)}$ is a shorthand for $\boldsymbol{\xi}$ and $\boldsymbol{\xi}^*$.

[4]Note that due to (17), the dual solution is invariant with respect to the transformation $\delta^{(*)} \to \delta^{(*)} + const.$ — such a transformation only adds a constant to the objective function, leaving the solution unaffected.

[5]subject to suitable conditions on $k$

[6]The present usage differs slightly from the standard definition of SVs, which are usually those that satisfy $\alpha_i^{(*)} > 0$. In our definition, SVs are those points where the constraints are active. However, the difference is marginal: (i) It follows from the KKT conditions that $\alpha_i^{(*)} > 0$ implies that the corresponding constraint is active. (ii) while it can happen in theory that a constraint is active and nevertheless the corresponding $\alpha_i^{(*)}$ is zero, this almost never occurs in practice.

[7]In the experiments, both the SVM optimization and the marching cubes rendering took up to about 2 hours.

## References

[1] J. Carr, R. Beatson, J. Cherrie, T. Mitchell, W. Fright, B. McCallum, and T. Evans. Reconstruction and representation of 3D objects with radial basis functions. In *Proc. 28th Ann. Conf. Computer Graphics and Interactive Techniques*, pages 67–76. 2001.

[2] C.-C. Chang and C.-J. Lin. *LIBSVM: a library for support vector machines*, 2001. Software available at `http://www.csie.ntu.edu.tw/~cjlin/libsvm`.

[3] O. Chapelle and B. Schölkopf. Incorporating invariances in nonlinear SVMs. In T.G. Dietterich, S. Becker, and Z. Ghahramani, editors, *Advances in Neural Information Processing Systems 14*, Cambridge, MA, 2002. MIT Press.

[4] J. Giesen and M. John. Surface reconstruction based on a dynamical system. *Computer Graphics Forum*, 21(3):363–371, 2002.

[5] T. Lewiner, H. Lopes, A. Wilson, and G. Tavares. Efficient implementation of marching cubes cases with topological guarantee. *Journal of Graphics Tools*, 8:1–15, 2003.

[6] S. Osher and N. Paragios. *Geometric Level Set Methods*. Springer, New York, 2003.

[7] M. Pauly, R. Keiser, and M. Gross. Multi-scale feature extraction on point-sampled surfaces. *Computer Graphics Forum*, 22(3):281–289, 2003.

[8] B. Schölkopf, J. Platt, J. Shawe-Taylor, A. J. Smola, and R. C. Williamson. Estimating the support of a high-dimensional distribution. *Neural Computation*, 13:1443–1471, 2001.

[9] I. Steinwart. Sparseness of support vector machines—some asymptotically sharp bounds. In S. Thrun, L. Saul, and B. Schölkopf, editors, *Advances in Neural Information Processing Systems 16*. MIT Press, Cambridge, MA, 2004.

[10] D. M. J. Tax and R. P. W. Duin. Support vector data description. *Machine Learning*, 54:45–66, 2004.

[11] V. N. Vapnik. *The Nature of Statistical Learning Theory*. Springer Verlag, New York, 1995.
